# Training Algorithms for Hidden Markov Models Using Entropy Based Distance Functions

Yoram Singer
AT&T Laboratories
600 Mountain Avenue
Murray Hill, NJ 07974
singer@research.att.com

Manfred K. Warmuth
Computer Science Department
University of California
Santa Cruz, CA 95064
manfred@cse.ucsc.edu

## Abstract

We present new algorithms for parameter estimation of HMMs. By adapting a framework used for supervised learning, we construct iterative algorithms that maximize the likelihood of the observations while also attempting to stay "close" to the current estimated parameters. We use a bound on the relative entropy between the two HMMs as a distance measure between them. The result is new iterative training algorithms which are similar to the EM (Baum-Welch) algorithm for training HMMs. The proposed algorithms are composed of a step *similar* to the expectation step of Baum-Welch and a new update of the parameters which replaces the maximization (re-estimation) step. The algorithm takes only negligibly more time per iteration and an approximated version uses the same expectation step as Baum-Welch. We evaluate experimentally the new algorithms on synthetic and natural speech pronunciation data. For sparse models, i.e. models with relatively small number of non-zero parameters, the proposed algorithms require significantly fewer iterations.

## 1 Preliminaries

We use the numbers from 0 to $N$ to name the states of an HMM. State 0 is a special initial state and state $N$ is a special final state. Any state sequence, denoted by s, starts with the initial state but never returns to it and ends in the final state. Observations symbols are also numbers in $\{1, \ldots, M\}$ and observation sequences are denoted by x. A discrete output hidden Markov model (HMM) is parameterized by two matrices $\mathbf{A}$ and $\mathbf{B}$. The first matrix is of dimension $[N, N]$ and $a_{i,j}$ ($0 \leq i \leq N - 1, 1 \leq j \leq N$) denotes the probability of moving from state $i$ to state $j$. The second matrix is of dimension $[N + 1, M]$ and $b_{i,k}$ is the probability of outputting symbol $k$ at state $i$. The set of parameters of an HMM is denoted by $\theta = (\mathbf{A}, \mathbf{B})$. (The initial state distribution vector is represented by the first row of $\mathbf{A}$.)

An HMM is a probabilistic generator of sequences. It starts in the initial state 0. It then iteratively does the following until the final state is reached. If $i$ is the current state then a next state $j$ is chosen according to the transition probabilities out of the current state (row $i$ of matrix $\mathbf{A}$). After arriving at state $j$ a symbol is output according to the output probabilities of that state (row $j$ of matrix $\mathbf{B}$). Let $P(\mathbf{x}, \mathbf{s}|\theta)$ denote the probability (likelihood) that an HMM $\theta$ generates the observation sequence x on the path s starting at state 0 and ending at state $N$: $P(\mathbf{x}, \mathbf{s} | |\mathbf{s}| = |\mathbf{x}| + 1, s_0 = 0, s_{|\mathbf{s}|} = N, \boldsymbol{\theta}) \stackrel{\text{def}}{=} \prod_{t=1}^{|\mathbf{x}|} a_{s_{t-1}, s_t} b_{s_t, x_t}$. For the sake of brevity we omit the conditions on s and x. Throughout the paper we assume that the HMMs are *absorbing*, that is from every state there is a path to the final state with a

non-zero probability. Similar parameter estimation algorithms can be derived for ergodic HMMs. Absorbing HMMs induce a probability over all state-observation sequences, i.e. $\sum_{\mathbf{x},\mathbf{s}} P(\mathbf{x},\mathbf{s}|\theta) = 1$. The likelihood of an observation sequence $\mathbf{x}$ is obtained by summing over all possible hidden paths (state sequences), $P(\mathbf{x}|\theta) = \sum_{\mathbf{s}} P(\mathbf{x},\mathbf{s}|\theta)$. To obtain the likelihood for a set $\mathcal{X}$ of observations we simply multiply the likelihood values for the individual sequences. We seek an HMM $\theta$ that maximizes the likelihood for a given set of observations $\mathcal{X}$, or equivalently, maximizes the log-likelihood, $LL(\mathcal{X}|\theta) = \frac{1}{|\mathcal{X}|} \sum_{\mathbf{x} \in \mathcal{X}} \ln P(\mathbf{x}|\theta)$.

To simplify our notation we denote the generic parameter in $\theta$ by $\theta_i$, where $i$ ranges from 1 to the total number of parameters in $\mathbf{A}$ and $\mathbf{B}$ (There might be less if some are clamped to zero). We denote the total number of parameters of $\theta$ by $\mathcal{I}$ and leave the (fixed) correspondence between the $\theta_i$ and the entries of $\mathbf{A}$ and $\mathbf{B}$ unspecified. The indices are naturally partitioned into classes corresponding to the rows of the matrices. We denote by $[i]$ the class of parameters to which $\theta_i$ belongs and by $\theta_{[i]}$ the vector of all $\theta_j$ s.t. $j \in [i]$. If $j \in [i]$ then both $\theta_i$ and $\theta_j$ are parameters from the same row of one of the two matrices. Whenever it is clear from the context, we will use $[i]$ to denote both a class of parameters and the row number (i.e. state) associated with the class. We now can rewrite $P(\mathbf{x},\mathbf{s}|\theta)$ as $\prod_{i=1}^{\mathcal{I}} \theta_i^{n_i(\mathbf{x},\mathbf{s})}$, where $n_i(\mathbf{x},\mathbf{s})$ is the number of times parameter $i$ is used along the path $\mathbf{s}$ with observation sequence $\mathbf{x}$. (Note that this value does not depend on the actual parameters $\theta$.) We next compute partial derivatives of the likelihood and the log-likelihood using this notation.

$$\frac{\partial}{\partial\theta_i} P(\mathbf{x},\mathbf{s}|\theta) = \theta_1^{n_1(\mathbf{x},\mathbf{s})} \cdots \theta_{i-1}^{n_{i-1}(\mathbf{x},\mathbf{s})} \, n_i(\mathbf{x},\mathbf{s}) \, \theta_i^{n_i(\mathbf{x},\mathbf{s})-1} \cdots \theta_I^{n_I(\mathbf{x},\mathbf{s})}$$

$$= \frac{n_i(\mathbf{x},\mathbf{s})}{\theta_i} \prod_{i=1}^{\mathcal{I}} \theta_i^{n_i(\mathbf{x},\mathbf{s})} = \frac{n_i(\mathbf{x},\mathbf{s})}{\theta_i} P(\mathbf{x},\mathbf{s}|\theta). \tag{1}$$

$$\frac{\partial LL(\mathcal{X}|\theta)}{\partial\theta_i} = \frac{1}{|\mathcal{X}|} \sum_{\mathbf{x} \in \mathcal{X}} \sum_{\mathbf{s}} \frac{\frac{\partial}{\partial\theta_i} P(\mathbf{x},\mathbf{s}|\theta)}{P(\mathbf{x}|\theta)} = \frac{1}{|\mathcal{X}|} \sum_{\mathbf{x} \in \mathcal{X}} \sum_{\mathbf{s}} \frac{n_i(\mathbf{x},\mathbf{s})}{\theta_i} \frac{P(\mathbf{x},\mathbf{s}|\theta)}{P(\mathbf{x}|\theta)}$$

$$= \frac{1}{|\mathcal{X}|} \sum_{\mathbf{x} \in \mathcal{X}} \sum_{\mathbf{s}} \frac{n_i(\mathbf{x},\mathbf{s})}{\theta_i} P(\mathbf{s}|\mathbf{x},\theta) = \frac{\sum_{\mathbf{x} \in \mathcal{X}} \hat{n}_i(\mathbf{x}|\theta)}{|\mathcal{X}|\theta_i}. \tag{2}$$

Here $\hat{n}_i(\mathbf{x}|\theta) \stackrel{\text{def}}{=} \sum_{\mathbf{s}} n_i(\mathbf{x},\mathbf{s})P(\mathbf{s}|\mathbf{x},\theta)$ is the expected number of occurrences of the transition/output that corresponds to $\theta_i$ over all paths that produce $\mathbf{x}$ in $\theta$. These values are calculated in the expectation step of the Expectation-Maximization (EM) training algorithm for HMMs [7], also known as the Baum-Welch [2] or the Forward-Backward algorithm. In the next sections we use the additional following expectations, $\hat{n}_i(\theta) \stackrel{\text{def}}{=} \sum_{\mathbf{x},\mathbf{s}} n_i(\mathbf{x},\mathbf{s})P(\mathbf{x},\mathbf{s}|\theta)$ and $\hat{n}_{[i]}(\theta) \stackrel{\text{def}}{=} \sum_{j \in [i]} \hat{n}_j(\theta)$. Note that the summation here is over all legal $\mathbf{x}$ and $\mathbf{s}$ of arbitrary length and $\hat{n}_{[i]}(\theta)$ is the expected number of times the state $[i]$ was visited.

## 2  Entropic distance functions for HMMs

Our training algorithms are based on the following framework of Kivinen and Warmuth for motivating iterative updates [6]. Assume we have already done a number of iterations and our current parameters are $\theta$. Assume further that $\mathcal{X}$ is the set of observations to be processed in the current iteration. In the batch case this set never changes and in the on-line case $X$ is typically a single observation. The new parameters $\tilde{\theta}$ should stay close to $\theta$, which incorporates all the knowledge obtained in past iterations, but it should also maximize the log-likelihood on the current date set $\mathcal{X}$. Thus, instead of maximizing the log-likelihood we maximize, $U(\tilde{\theta}) = \eta LL(\mathcal{X}|\tilde{\theta}) - d(\tilde{\theta},\theta)$ (see [6, 5] for further motivation).

Here $d$ measures the distance between the old and new parameters and $\eta > 0$ is a trade-off factor. Maximizing $U(\tilde{\theta})$ is usually difficult since both the distance function and the log-likelihood depend on $\tilde{\theta}$. As in [6, 5], we approximate the log-likelihood by a first order Taylor expansion around $\tilde{\theta} = \theta$ and add Lagrange multipliers for the constraints that the parameters of each class must sum to one:

$$U(\tilde{\theta}) \approx \eta \left( LL(\mathcal{X}|\theta) + (\tilde{\theta} - \theta)\nabla_{\theta} LL(\mathcal{X}|\theta) \right) - d(\tilde{\theta}, \theta) + \sum_{[i]} \lambda_{[i]} \sum_{j \in [i]} \tilde{\theta}_j \ . \quad (3)$$

A commonly used distance function is the relative entropy. To calculate the relative entropy between two HMMs we need to sum over all possible hidden state sequence which leads to the following definition,

$$\mathrm{d}_{RE}(\tilde{\theta}, \theta) \overset{\text{def}}{=} \sum_{\mathbf{x}} P(\mathbf{x}|\tilde{\theta}) \ \ln \frac{P(\mathbf{x}|\tilde{\theta})}{P(\mathbf{x}|\theta)} = \sum_{\mathbf{x}} \left( \sum_{\mathbf{s}} P(\mathbf{x}, \mathbf{s}|\tilde{\theta}) \right) \ \ln \frac{\sum_{\mathbf{s}} P(\mathbf{x}, \mathbf{s}|\tilde{\theta})}{\sum_{\mathbf{s}} P(\mathbf{x}, \mathbf{s}|\theta)}$$

However, the above divergence is very difficult to calculate and is not a convex function in $\theta$. To avoid the computational difficulties and the non-convexity of $\mathrm{d}_{RE}$ we upper bound the relative entropy using the *log sum inequality* [3]:

$$
\begin{aligned}
\mathrm{d}_{RE}(\tilde{\theta}, \theta) \ &\leq \ \widehat{d}_{RE}(\tilde{\theta}, \theta) \overset{\text{def}}{=} \sum_{\mathbf{x}, \mathbf{s}} P(\mathbf{x}, \mathbf{s}|\tilde{\theta}) \ \ln \frac{P(\mathbf{x}, \mathbf{s}|\tilde{\theta})}{P(\mathbf{x}, \mathbf{s}|\theta)} \\
&= \ \sum_{\mathbf{x}, \mathbf{s}} P(\mathbf{x}, \mathbf{s}|\tilde{\theta}) \ \ln \left( \frac{\prod_{i=1}^{\mathcal{I}} \tilde{\theta}_i^{n_i(\mathbf{x}, \mathbf{s})}}{\prod_{i=1}^{\mathcal{I}} \theta_i^{n_i(\mathbf{x}, \mathbf{s})}} \right) = \sum_{\mathbf{x}, \mathbf{s}} P(\mathbf{x}, \mathbf{s}|\tilde{\theta}) \sum_{i=1}^{\mathcal{I}} n_i(\mathbf{x}, \mathbf{s}) \ \ln \frac{\tilde{\theta}_i}{\theta_i} \\
&= \ \sum_{i=1}^{\mathcal{I}} \ln \frac{\tilde{\theta}_i}{\theta_i} \sum_{\mathbf{x}, \mathbf{s}} P(\mathbf{x}, \mathbf{s}|\tilde{\theta}) \ n_i(\mathbf{x}, \mathbf{s}) = \sum_{i=1}^{\mathcal{I}} \hat{n}_i(\tilde{\theta}) \ln \frac{\tilde{\theta}_i}{\theta_i}
\end{aligned}
$$

Note that for the distance function $\widehat{d}_{RE}(\tilde{\theta}, \theta)$ an HMM is viewed as a joint distribution between observation sequences and hidden state sequences. We can further simplify the bound on the relative entropy using the following lemma (proof omitted).

**Lemma 1** *For any absorbing HMM, $\theta$, and any parameter $\theta_i \in \theta$, $\hat{n}_i(\theta) = \theta_i \hat{n}_{[i]}(\theta)$.*

This gives the following new formula, $\widehat{d}_{RE}(\tilde{\theta}, \theta) = \sum_{i=1}^{\mathcal{I}} \hat{n}_{[i]}(\tilde{\theta}) \left[ \tilde{\theta}_i \ln \frac{\tilde{\theta}_i}{\theta_i} \right]$, which can be rewritten as, $\widehat{d}_{RE}(\tilde{\theta}, \theta) = \sum_{[i]} \hat{n}_{[i]}(\tilde{\theta}) \, \mathrm{d}_{RE}(\tilde{\theta}_{[i]}, \theta_{[i]}) = \sum_{[i]} \hat{n}_{[i]}(\tilde{\theta}) \sum_{j \in [i]} \tilde{\theta}_j \ \ln \frac{\tilde{\theta}_i}{\theta_i}$ . Equation (3) is still difficult to solve since the variables $\hat{n}_{[i]}(\tilde{\theta})$ depend on the new set of parameters (which are not known). We therefore further approximate $\widehat{d}_{RE}(\tilde{\theta}, \theta)$ by the distance function, $\widehat{\widehat{d}}_{RE}(\tilde{\theta}, \theta) = \sum_{[i]} \hat{n}_{[i]}(\theta) \sum_{j \in [i]} \tilde{\theta}_j \ \ln \frac{\tilde{\theta}_i}{\theta_i}$.

## 3 New Parameter Updates

We now would like to use the distance functions discussed in previous section in $U(\tilde{\theta})$. We first derive our main update using this distance function. This is done by replacing $d(\tilde{\theta}, \theta)$ in $U(\tilde{\theta})$ with $\widehat{\widehat{d}}_{RE}(\tilde{\theta}, \theta)$ and setting the derivatives of the resulting $U(\tilde{\theta})$ w.r.t $\tilde{\theta}_i$ to 0. This gives the following set of equations ($i \in \{1, \ldots, \mathcal{I}\}$),

$$\eta \frac{\sum_{\mathbf{x} \in \mathcal{X}} \hat{n}_i(\mathbf{x}|\theta)}{|\mathcal{X}|\theta_i} - \hat{n}_{[i]}(\theta) \, (\ln \frac{\tilde{\theta}_i}{\theta_i} - 1) + \lambda_{[i]} = 0 \ ,$$

which are equivalent to

$$\frac{\eta}{\hat{n}_{[i]}(\theta)} \frac{\sum_{\mathbf{x} \in \mathcal{X}} \hat{n}_i(\mathbf{x}|\theta)}{|\mathcal{X}|\theta_i} - \ln \frac{\tilde{\theta}_i}{\theta_i} + \lambda'_{[i]} = 0 \ .$$

We now can solve for $\tilde{\theta}_i$ and replace $\lambda'_{[i]}$ by a normalization factor which ensures that the sum of the parameters in $[i]$ is 1:

$$\tilde{\theta}_i = \frac{\theta_i \exp\left(\frac{\eta}{\hat{n}_{[i]}(\theta)} \frac{\sum_{\mathbf{x} \in \mathcal{X}} \hat{n}_i(\mathbf{x}|\theta)}{|\mathcal{X}| \theta_i}\right)}{\sum_{j \in [i]} \theta_j \exp\left(\frac{\eta}{\hat{n}_{[i]}(\theta)} \frac{\sum_{\mathbf{x} \in \mathcal{X}} \hat{n}_j(\mathbf{x}|\theta)}{|\mathcal{X}| \theta_j}\right)} \ . \tag{4}$$

The above re-estimation rule is the *entropic update* for HMMs.[1]

We now derive an alternate to the update of (4). The mixture weights $\hat{n}_{[i]}(\theta)$ (which approximate the original mixture weights $\hat{n}_{[i]}(\tilde{\theta})$ in $\hat{d}_{RE}(\tilde{\theta}, \theta))$ lead to a state dependent learning rate of $\frac{\eta}{\hat{n}_{[i]}(\theta)}$ for the parameters of class $[i]$. If computation time is limited (see discussion below) then the expectations $\hat{n}_{[i]}(\theta)$ can be approximated by values that are readily available. One possible choice is to use the sample based expectations $\sum_{j \in [i]} \sum_{\mathbf{x} \in \mathcal{X}} \hat{n}_j(\mathbf{x}|\theta)/|\mathcal{X}|$ as an approximation for $\hat{n}_{[i]}(\theta)$. These weights are needed for calculating the gradient and are evaluated in the expectation step of Baum-Welch. Let, $\hat{n}_{[i]}(\mathbf{x}|\theta) \overset{\text{def}}{=} \sum_{j \in [i]} \hat{n}_j(\mathbf{x}|\theta)$, then this approximation leads to the following distance function

$$\sum_{[i]} \frac{\sum_{\mathbf{x} \in \mathcal{X}} \hat{n}_{[i]}(\mathbf{x}|\theta)}{|\mathcal{X}|} d_{RE}(\tilde{\theta}_{[i]}, \theta_{[i]}) = \sum_{[i]} \frac{\sum_{\mathbf{x} \in \mathcal{X}} \hat{n}_{[i]}(\mathbf{x}|\theta)}{|\mathcal{X}|} \sum_{j \in [i]} \tilde{\theta}_j \ln \frac{\tilde{\theta}_j}{\theta_j} \ , \tag{5}$$

which results in an update which we call the *approximated entropic update* for HMMs:

$$\tilde{\theta}_i = \frac{\theta_i \exp\left(\frac{\eta}{\sum_{\mathbf{x} \in \mathcal{X}} \hat{n}_{[i]}(\mathbf{x}|\theta)} \frac{\sum_{\mathbf{x} \in \mathcal{X}} \hat{n}_i(\mathbf{x}|\theta)}{\theta_i}\right)}{\sum_{j \in [i]} \theta_j \, exp\left(\frac{\eta}{\sum_{\mathbf{x} \in \mathcal{X}} \hat{n}_{[i]}(\mathbf{x}|\theta)} \frac{\sum_{\mathbf{x} \in \mathcal{X}} \hat{n}_j(\mathbf{x}|\theta)}{\theta_j}\right)} \ . \tag{6}$$

Given a current set of parameters $\theta$ and a learning rate $\eta$ we obtain a new set of parameters $\tilde{\theta}$ by iteratively evaluating the right-hand-side of the entropic update or the approximated entropic update. We calculate the expectations $\hat{n}_i(\mathbf{x}|\theta)$ as done in the expectation step of Baum-Welch. The weights $\hat{n}_{[i]}(\mathbf{x}|\theta)$ are obtained by averaging $\hat{n}_j(\mathbf{x}|\theta)$ for $j \in [i]$. This lets us evaluate the right-hand-side of the approximated entropic update. The entropic update is slightly more involved and requires an additional calculation of $\hat{n}_{[i]}(\theta)$. (Recall that $\hat{n}_{[i]}(\theta)$ is the expected number of times state $[i]$ is visited, *unconditioned* on the data). To compute these expectations we need to sum over all possible sequences of state-observation pairs. Since the probability of outputting the possible symbols at a given state sum to one, calculating $\hat{n}_{[i]}(\theta)$ reduces to evaluating the probability of reaching a state for each possible time and sequence length. For absorbing HMMs $\hat{n}_{[i]}(\theta)$ can be approximated efficiently using dynamic programming; we compute $\hat{n}_{[i]}(\theta)$ by summing the probabilities of all legal state sequences s of up to length $CN$ (typically $C = 3$ proved to be sufficient to obtain very accurate approximations of $\hat{n}_{[i]}(\theta)$). Therefore, the time complexity of calculating $\hat{n}_{[i]}(\theta)$ depends only on the number of states, regardless of the dimension of the output vector $M$ and the training data $\mathcal{X}$.

## 4 The relation to EM and convergence properties

We first show that the EM algorithm for HMMs can be derived using our framework. To do so, we approximate the relative entropy by the $\chi^2$ distance (see [3]), $d_{RE}(\tilde{p}, p) \approx d_{\chi^2}(\tilde{p}, p) \stackrel{def}{=} \frac{1}{2} \sum_i \frac{(\tilde{p}_i - p_i)^2}{p_i}$, and use this distance to approximate $\widehat{d}_{RE}(\tilde{\theta}, \theta)$:

$$\widehat{d}_{RE}(\tilde{\theta}, \theta) \approx \widehat{d}_{\chi^2}(\tilde{\theta}, \theta) \stackrel{def}{=} \sum_{[i]} \hat{n}_{[i]}(\tilde{\theta}) \ d_{\chi^2}(\tilde{\theta}_{[i]}, \theta_{[i]})$$

$$\approx \sum_{[i]} \hat{n}_{[i]}(\theta) \ d_{\chi^2}(\tilde{\theta}_{[i]}, \theta_{[i]}) \approx \sum_{[i]} \frac{\sum_{\mathbf{x} \in \mathcal{X}} \hat{n}_{[i]}(\mathbf{x}|\theta)}{|\mathcal{X}|} \ d_{\chi^2}(\tilde{\theta}_{[i]}, \theta_{[i]}) \ .$$

Here $d_{\chi^2}(\tilde{\theta}_{[i]}, \theta_{[i]}) = \frac{1}{2} \sum_{j \in [i]} \frac{(\tilde{\theta}_i - \theta_i)^2}{\theta_i}$. By minimizing $U(\tilde{\theta})$ with the last version of the $\chi^2$ distance function and following the same derivation steps as for the approximated entropic update we arrive at what we call the *approximated $\chi^2$ update* for HMMs:

$$\tilde{\theta}_i = (1 - \eta)\theta_i + \eta \sum_{\mathbf{x} \in \mathcal{X}} \hat{n}_i(\mathbf{x}|\theta) \Big/ \sum_{\mathbf{x} \in \mathcal{X}} \hat{n}_{[i]}(\mathbf{x}|\theta) \ . \tag{7}$$

Setting $\eta = 1$ results in the update, $\tilde{\theta}_i = \sum_{\mathbf{x} \in \mathcal{X}} \hat{n}_i(\mathbf{x}|\theta) / \sum_{\mathbf{x} \in \mathcal{X}} \hat{n}_{[i]}(\mathbf{x}|\theta)$, which is the maximization (re-estimation) step of the EM algorithm.

Although omitted from this paper due to the lack of space, it is can be shown that for $\eta \in (0, 1]$ the entropic updates and the $\chi^2$ update improve the likelihood on each iteration. Therefore, these updates belong to the family of Generalized EM (GEM) algorithms which are guaranteed to converge to a local maximum given some additional conditions [4]. Furthermore, using infinitesimal analysis and second order approximation of the likelihood function at the (local) maximum similar to [10], it can be shown that the approximated $\chi^2$ update is a contraction mapping and close to the local maximum there exists a learning rate $\eta > 1$ which results in a faster rate of convergence than when using $\eta = 1$.

## 5 Experiments with Artificial and Natural Data

In order to test the actual convergence rate of the algorithms and to compare them to Baum-Welch we created synthetic data using HMMs. In our experiments we mainly used sparse models, that is, models with many parameters clamped to zero. Previous work (e.g., [5, 6]) might suggest that the entropic updates will perform better on sparse models. (Indeed, when we used dense models to generate the data, the algorithms showed almost the same performance). The training algorithms, however, were started from a randomly chosen *dense* model. When comparing the algorithms we used the same initial model. Due to different trajectories in parameter space, each algorithm may converge to a different (local) maximum. For the clarity of presentation we show here results for cases where all updates converged to the same maximum, which often occur when the HMM generating the data is sparse and there are enough examples (typically tens of observations per non-zero parameter). We tested both the entropic updates and the $\chi^2$ updates. Learning rates greater than one speed up convergence. The two entropic updates converge almost equally fast on synthetic data generated by an HMM. For natural data the entropic update converges slightly faster than the approximated version. The $\chi^2$ update also benefits from learning rates larger than one. However, the $\chi^2$-update need to be used carefully since it does not necessarily ensure non-negativeness of the new parameters for $\eta > 1$. This problems is exaggerated when the data is not generated by an HMM. We therefore used the entropic updates in our experiments with natural data. In order to have a fair comparison, we did *not* tune the learning rate $\eta$ and set it to 1.5. In Figure 1 we give a comparison of the entropic update, the approximated entropic update, and Baum-Welch (left figure), using an HMM to generate the random observation sequences, where $N = M = 40$ but only 25% (10 parameters on the average for each transition/observation vector) of the parameters of the

HMM are non-zero. The performance of the entropic update and the approximated entropic update are practically the same and both updates clearly outperform Baum-Welch. One reason the performance of the two entropic updates is the same is that the observations were indeed generated by an HMM. In this case, approximating the expectations $\hat{n}_{[i]}(\theta)$ by the sample based expectations seems reasonable. These results suggest a valuable alternative to using Baum-Welch with a *predetermined* sparse, potentially biased, HMM where a large number of parameters is clamped to zero. Instead, we suggest starting with a full model and let one of the entropic updates find the relevant parameters. This approach is demonstrated on the right part of Figure 1. In this example the data was generated by a sparse HMM with 100 states and 100 possible output symbols. Only 10% of the HMM's parameters were non-zero. Three log-likelihood curves are given in the figure. One is the log-likelihood achieved by Baum-Welch when only those parameters that are non-zero in the HMM generating the data are initialized to random non-zero values. The other two are the log-likelihood of the entropic update and Baum-Welch when *all* the parameters are initialized randomly. The curves show that the entropic update compensates for its inferior initialization in less than 10 iterations (see horizontal line in Figure 1) and from this point on it requires only 23 more iterations to converge compared to Baum-Welch which is given prior knowledge of the non-zero parameters. In contrast, when Baum-Welch is started with a full model then its convergence is much slower than the entropic update.

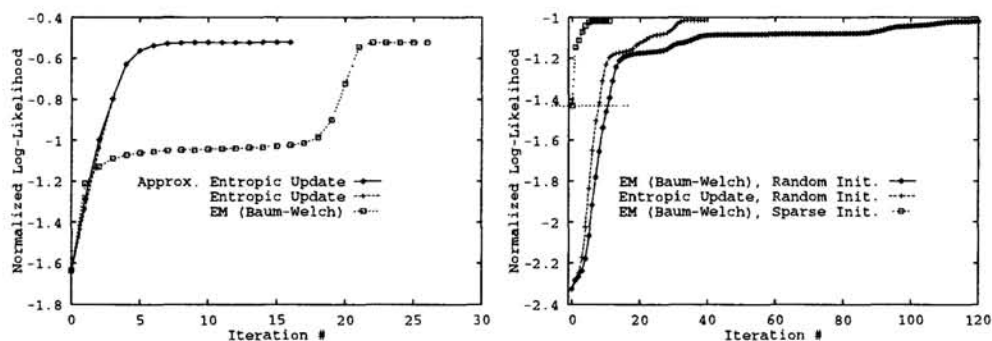

Figure 1: Comparison of the entropic updates and Baum-Welch.

We next tested the updates on speech pronunciation data. In natural speech, a word might be pronounced differently by different speakers. A common practice is to construct a set of stochastic models in order to capture the variability of the possible pronunciations. alternative pronunciations of a given word. This problem was studied previously in [9] using a state merging algorithm for HMMs and in [8] using a subclass of probabilistic finite automata. The purpose of the experiments discussed here is not to compare the above algorithms to the entropic updates but rather compare the entropic updates to Baum-Welch. Nevertheless, the resulting HMM pronunciation models are usually sparse. Typically, only two or three phonemes have a non zero output probability at a given state and the average number of states that in practice can follow a states is about 2. Therefore, the entropic updates may provide a good alternative to the algorithms presented in [8, 9].

We used the TIMIT (Texas Instruments-MIT) database as in [8, 9]. This database contains the acoustic waveforms of continuous speech with phone labels from an alphabet of 62 phones which constitute a temporally aligned phonetic transcription to the uttered words. For the purpose of building pronunciation models, the acoustic data was ignored and we partitioned the phonetic labels according to the words that appeared in the data. The data was filtered and partitioned so that words occurring between 20 and 100 times in the dataset were used for training and evaluation according to the following partition. 75% of the occurrences of each word were used as training data for the learning algorithm and the remaining 25% were used for evaluation. We then built for each word three pronunciation models by training a fully connected HMM whose number of states was set to 1, 1.5 and 1.75 times the longest sample (denoted by $N_m$). The models were evaluated by calculating

the log-likelihood (averaged over 10 different random parameter initializations) of each HMM on the phonetic transcription of each word in the test set. In Table 1 we give the negative log-likelihood achieved on the test data together with the average number of iterations needed for training. Overall the differences in the log-likelihood are small which means that the results should be interpreted with some caution. Nevertheless, the entropic update obtained the highest likelihood on the test data while needing the least number of iterations. The approximated entropic update and Baum-Welch achieve similar results on the test data but the latter requires more iterations. Checking the resulting models reveals one reason why the entropic update achieves higher likelihood values, namely, it does a better job in setting the irrelevant parameters to zero (and it does it faster).

| | Negative Log-Likelihood | | | # Iterations | | |
|---|---|---|---|---|---|---|
| # States | $1.0N_m$ | $1.5N_m$ | $1.75N_m$ | $1.0N_m$ | $1.5N_m$ | $1.75N_m$ |
| Baum-Welch | 2448 | 2388 | 2425 | 27.4 | 36.1 | 41.1 |
| Approx. EU | 2440 | 2389 | 2426 | 25.5 | 35.0 | 37.0 |
| Entropic Update | 2418 | 2352 | 2405 | 23.1 | 30.9 | 32.6 |

Table 1: Comparison of the entropic updates and Baum-Welch on speech pronunciation data.

# 6 Conclusions and future research

In this paper we have showed how the framework of Kivinen and Warmuth [6] can be used to derive parameter updates algorithms for HMMs. We view an HMM as a joint distribution between the observation sequences and hidden state sequences and use a bound on relative entropy as a distance between the new and old parameter settings. If we approximate of the relative entropy by the $\chi^2$ distance, replace the exact state expectations by a sample based approximation, and fix the learning rate to one then the framework yields an alternative derivation of the EM algorithm for HMMs. Since the EM update uses sample based estimates of the state expectations it is hard to use it in an on-line setting. In contrast, the on-line versions of our updates can be easily derived using only one observation sequence at a time. Also, there are alternative gradient descent based methods for estimating the parameters of HMMs. Such methods usually employ an exponential parameterization (such as soft-max) of the parameters (see [1]). For the case of learning one set of mixture coefficients an exponential parameterization led to an algorithm with a slower convergence rate compared to algorithms derived using entropic distances [5]. However, it is not clear whether this is still the case for HMMs. Our future goals is to perform a comparative study of the different updates with emphasis on the on-line versions.

**Acknowledgments**
We thank Anders Krogh for showing us the simple derivative calculations used in this paper and thank Fernando Pereira and Yasubumi Sakakibara for interesting discussions.

## Footnotes

[1] A subtle improvement is possible over the update (4) by treating the transition probabilities and output probabilities differently. First the transition probabilities are updated based on (4). Then the state probabilities $\hat{n}_{[i]}(\tilde{\theta}) = \hat{n}_{[i]}(\tilde{\mathbf{A}})$ are recomputed based on the new parameters $\tilde{\mathbf{A}}$. This is possible since the state probabilities depend only on the transition probabilities and not on the output probabilities. Finally the output probabilities are updated with (4) where the $\hat{n}_{[i]}(\tilde{\theta})$ are used in place of the $\hat{n}_{[i]}(\theta)$.

# References

[1] P. Baldi and Y. Chauvin. Smooth on-line learning algorithms for Hidden Markov Models. *Neural Computation*, 6(2), 1994.

[2] L.E. Baum and T. Petrie. Statistical inference for probabilistic functions of finite state markov chains. *Annals of Mathematical Statisitics*, 37, 1966.

[3] T. Cover and J. Thomas. *Elements of Information Theory*. Wiley, 1991.

[4] A. P. Dempster, N. M. Laird, and D. B. Rubin. Maximum-likelihood from incomplete data via the EM algorithm. *Journal of the Royal Statistical Society*, B39:1–38, 1977.

[5] D. P. Helmbold, R. E. Schapire, Y. Singer, and M. K. Warmuth. A comparison of new and old algorithms for a mixture estimation problem. In *Proceedings of the Eighth Annual Workshop on Computational Learning Theory*, pages 69–78, 1995.

[6] J. Kivinen and M. K. Warmuth. Exponentiated gradient versus gradient descent for linear predictors. *Informationa and Computation*, 1997. To appear.

[7] L.R. Rabiner and B. H. Juang. An introduction to hidden markov models. *IEEE ASSP Magazine*, 3(1):4–16, 1986.

[8] D. Ron, Y. Singer, and N. Tishby. On the learnability and usage of acyclic probabilistic finite automata. In *Proc. of the Eighth Annual Workshop on Computational Learning Theory*, 1995.

[9] A. Stolcke and S. Omohundro. Hidden Markov model induction by Bayesian model merging. In *Advances in Neural Information Processing Systems*, volume 5. Morgan Kaufmann, 1993.

[10] L. Xu and M.I. Jordan. On convergence properties of the EM algorithm for Gaussian mixtures. *Neural Computation*, 8:129–151, 1996.